# A Probabilistic Model for Online Document Clustering with Application to Novelty Detection

**Jian Zhang**†
†School of Computer Science
Cargenie Mellon University
Pittsburgh, PA 15213
jian.zhang@cs.cmu.edu

**Zoubin Ghahramani**††‡
‡ Gatsby Computational Neuroscience Unit
University College London
London WC1N 3AR, UK
zoubin@gatsby.ucl.ac.uk

**Yiming Yang**†
†School of Computer Science
Cargenie Mellon University
Pittsburgh, PA 15213
yiming@cs.cmu.edu

## Abstract

In this paper we propose a probabilistic model for online document clustering. We use non-parametric Dirichlet process prior to model the growing number of clusters, and use a prior of general English language model as the base distribution to handle the generation of novel clusters. Furthermore, cluster uncertainty is modeled with a Bayesian Dirichlet-multinomial distribution. We use empirical Bayes method to estimate hyperparameters based on a historical dataset. Our probabilistic model is applied to the novelty detection task in Topic Detection and Tracking (TDT) and compared with existing approaches in the literature.

## 1 Introduction

The task of online document clustering is to group documents into clusters as long as they arrive in a temporal sequence. Generally speaking, it is difficult for several reasons: First, it is unsupervised learning and the learning has to be done in an online fashion, which imposes constraints on both strategy and efficiency. Second, similar to other learning problems in text, we have to deal with a high-dimensional space with tens of thousands of features. And finally, the number of clusters can be as large as thousands in newswire data.

The objective of novelty detection is to identify the novel objects from a sequence of data, where "novel" is usually defined as dissimilar to previous seen instances. Here we are interested in novelty detection in the text domain, where we want to identify the earliest report of every new event in a sequence of news stories. Applying online document clustering to the novelty detection task is straightforward by assigning the first seed of every cluster as novel and all its remaining ones as non-novel. The most obvious application of novelty detection is that, by detecting novel events, systems can automatically alert people when new events happen, for example.

In this paper we apply Dirichlet process prior to model the growing number of clusters, and propose to use a General English language model as a basis of newly generated clusters. In particular, the new clusters will be generated according to the prior and a background General English model, and each document cluster is modeled using a Bayesian Dirichlet-multinomial language model. The Bayesian inference can be easily carried out due to conjugacy, and model hyperparameters are estimated using a historical dataset by the empirical Bayes method. We evaluate our online clustering algorithm (as well as its variants) on the novelty detection task in TDT, which has been regarded as the hardest task in that literature [2].

The rest of this paper is organized as follows. We first introduce our probabilistic model in Section 2, and in Section 3 we give detailed information on how to estimate model hyperparameters. We describe the experiments in Section 4, and related work in Section 5. We conclude and discuss future work in Section 6.

## 2  A Probabilistic Model for Online Document Clustering

In this section we will describe the generative probabilistic model for online document clustering. We use $\mathbf{x} = (n_1^{(\mathbf{x})}, n_2^{(\mathbf{x})}, \ldots, n_V^{(\mathbf{x})})$ to represent a document vector where each element $n_v^{(\mathbf{x})}$ denotes the term frequency of the $v^{th}$ corresponding word in the document $\mathbf{x}$, and $V$ is the total size of the vocabulary.

### 2.1  Dirichlet-Multinomial Model

The multinomial distribution has been one of the most frequently used language models for modeling documents in information retrieval. It assumes that given the set of parameters $\theta = (\theta_1, \theta_2, \ldots, \theta_V)$, a document $\mathbf{x}$ is generated with the following probability:

$$p(\mathbf{x}|\theta) = \frac{(\sum_{v=1}^{V} n_v^{(\mathbf{x})})!}{\prod_{v=1}^{V} n_v^{(\mathbf{x})}!} \prod_{v=1}^{V} \theta_v^{n_v^{(\mathbf{x})}}.$$

From the formula we can see the so-called naive assumption: words are assumed to be independent of each other. Given a collection of documents generated from the same model, the parameter $\theta$ can be estimated with Maximum Likelihood Estimation (MLE).

In a Bayesian approach we would like to put a Dirichlet prior over the parameter ($\theta \sim Dir(\alpha)$) such that the probability of generating a document is obtained by integrating over the parameter space: $p(\mathbf{x}) = \int p(\theta|\alpha)p(\mathbf{x}|\theta)d\theta$. This integration can be easily written down due to the conjugacy between Dirichlet and multinomial distributions. The key difference between the Bayesian approach and the MLE is that the former uses a distribution to model the uncertainty of the parameter $\theta$, while the latter gives only a point estimation.

### 2.2  Online Document Clustering with Dirichlet Process Mixture Model

In our system documents are grouped into clusters in an online fashion. Each cluster is modeled with a multinomial distribution whose parameter $\theta$ follows a Dirichlet prior. First, a cluster is chosen based on a Dirichlet process prior (can be either a new or existing cluster), and then a document is drawn from that cluster.

We use Dirichlet Process (DP) to model the prior distribution of $\theta$'s, and our hierarchical model is as follows:

$$\begin{aligned}
\mathbf{x}_i|c_i &\sim Mul(.|\theta^{(c_i)}) \\
\theta^i &\overset{iid.}{\sim} G \\
G &\sim DP(\lambda, G_0)
\end{aligned} \tag{1}$$

where $c_i$ is the cluster indicator variable, $\theta^i$ is the multinomial parameter [1] for each document, and $\theta^{(c_i)}$ is the unique $\theta$ for the cluster $c_i$. $G$ is a random distribution generated from the Dirichlet process $DP(\lambda, G_0)$ [4], which has a precision parameter $\lambda$ and a base distribution $G_0$. Here our base distribution $G_0$ is a Dirichlet distribution $Dir(\gamma\pi_1, \gamma\pi_2, \ldots, \gamma\pi_V)$ with $\sum_{t=1}^{V} \pi_t = 1$, which reflects our expected knowledge about $G$. Intuitively, our $G_0$ distribution can be treated as the prior over general English word frequencies, which has been used in information retrieval literature [6] to model general English documents.

The exact cluster-document generation process can be described as follows:

1. Let $\mathbf{x}_i$ be the current document under processing (the $i^{th}$ document in the input sequence), and $\mathbf{C}_1, \mathbf{C}_2, \ldots, \mathbf{C}_m$ are already generated clusters.

2. Draw a cluster $c_i$ based on the following Dirichlet process prior [4]:

$$
\begin{aligned}
p(c_i = \mathbf{C}_j) &= \frac{|\mathbf{C}_j|}{\lambda + \sum_{j=1}^{m} |\mathbf{C}_j|} \quad (j = 1, 2, \ldots, m) \\
p(c_i = \mathbf{C}_{m+1}) &= \frac{\lambda}{\lambda + \sum_{j=1}^{m} |\mathbf{C}_j|}
\end{aligned}
\tag{2}
$$

where $|\mathbf{C}_j|$ stands for the cardinality of cluster $j$ with $\sum_{j=1}^{m} |\mathbf{C}_j| = i - 1$, and with certain probability a new cluster $\mathbf{C}_{m+1}$ will be generated.

3. Draw the document $\mathbf{x}_i$ from the cluster $c_i$.

## 2.3 Model Updating

Our models for each cluster need to be updated based on incoming documents. We can write down the probability that the current document $\mathbf{x}_i$ is generated by any cluster as

$$
p(\mathbf{x}_i | \mathbf{C}_j) = \int p(\theta^{(\mathbf{C}_j)} | \mathbf{C}_j) p(\mathbf{x}_i | \theta^{(\mathbf{C}_j)}) d\theta^{(\mathbf{C}_j)} \quad (j = 1, 2, \ldots, m, m+1)
$$

where $p(\theta^{(\mathbf{C}_j)} | \mathbf{C}_j)$ is the posterior distribution of parameters of the $j^{th}$ cluster ($j = 1, 2, \ldots, m$) and we use $p(\theta^{(\mathbf{C}_{m+1})} | \mathbf{C}_{m+1}) = p(\theta^{(\mathbf{C}_{m+1})})$ to represent the prior distribution of the parameters of the new cluster for convenience. Although the dimensionality of $\theta$ is high ($V \approx 10^5$ in our case), closed-form solution can be obtained under our Dirichlet-multinomial assumption. Once the conditional probabilities $p(\mathbf{x}_i | \mathbf{C}_j)$ are computed, the probabilities $p(\mathbf{C}_j | \mathbf{x}_i)$ can be easily calculated using the Bayes rule:

$$
p(\mathbf{C}_j | \mathbf{x}_i) = \frac{p(\mathbf{C}_j) p(\mathbf{x}_i | \mathbf{C}_j)}{\sum_{j'=1}^{m+1} p(\mathbf{C}_{j'}) p(\mathbf{x}_i | \mathbf{C}_{j'})}
$$

where the prior probability of each cluster is calculated using equation (2).

Now there are several choices we can consider on how to update the cluster models. The first choice, which is correct but obviously intractable, is to fork $m + 1$ children of the current system where the $j^{th}$ child is updated with document $\mathbf{x}_i$ assigned to cluster $j$, while the final system is a probabilistic combination of those children with the corresponding probabilities $p(\mathbf{C}_j | \mathbf{x}_i)$. The second choice is to make a hard decision by assigning the current document $\mathbf{x}_i$ to the cluster with the maximum probability:

$$
\mathbf{c}_i = \arg \max_{\mathbf{C}_j} p(\mathbf{C}_j | \mathbf{x}_i) = \frac{p(\mathbf{C}_j) p(\mathbf{x}_i | \mathbf{C}_j)}{\sum_{j'=1}^{m+1} p(\mathbf{C}_{j'}) p(\mathbf{x}_i | \mathbf{C}_{j'})}.
$$

The third choice is to use a soft probabilistic updating, which is similar in spirit to the Assumed Density Filtering (ADF) [7] in the literature. That is, each cluster is updated by exponentiating the likelihood function with probabilities:

$$p(\theta^{(\mathbf{C}_j)}|\mathbf{x}_i, \mathbf{C}_j) \quad \propto \quad \left(p(\mathbf{x}_i|\theta^{(\mathbf{C}_j)})\right)^{p(\mathbf{C}_j|\mathbf{x}_i)} p(\theta^{(\mathbf{C}_j)}|\mathbf{C}_j)$$

However, we have to specially deal with the new cluster since we cannot afford both time-wise and space-wise to generate a new cluster for each incoming document. Instead, we will update all existing clusters as above, and new cluster will be generated only if $c_i = \mathbf{C}_{m+1}$. We will use HD and PD (hard decision and probabilistic decision) to denote the last two candidates in our experiments.

## 3 Learning Model Parameters

In the above probabilistic model there are still several hyperparameters not specified, namely the $\pi$ and $\gamma$ in the base distribution $G_0 = Dir(\gamma\pi_1, \gamma\pi_2, \ldots, \gamma\pi_V)$, and the precision parameter $\lambda$ in the $DP(\lambda, G_0)$. Since we can obtain a partially labeled historical dataset [2], we now discuss how to estimate those parameters respectively.

We will mainly use the empirical Bayes method [5] to estimate those parameters instead of taking a full Bayesian approach, since it is easier to compute and generally reliable when the number of data points is relatively large compared to the number of parameters. Because the $\theta^i$'s are iid. from the random distribution $G$, by integrating out the $G$ we get

$$\theta^i|\theta^1, \theta^2, \ldots, \theta^{i-1} \sim \frac{\lambda}{\lambda + i - 1}G_0 + \frac{1}{\lambda + i - 1}\sum_{j<i}\delta_{\theta^j}$$

where the distribution is a mixture of continuous and discrete distributions, and the $\delta_\theta$ denotes the probability measure giving point mass to $\theta$.

Now suppose we have a historical dataset $H$ which contains $K$ labeled clusters $H_j(j = 1, 2, \ldots, K)$, with the $k^{th}$ cluster $H_k = \{\mathbf{x}_{k,1}, \mathbf{x}_{k,2}, \ldots, \mathbf{x}_{k,m_k}\}$ having $m_k$ documents. The joint probability of $\theta$'s of all documents can be obtained as

$$p(\theta^1, \theta^2, \ldots, \theta^{|H|}) = \prod_{i=1}^{|H|}(\frac{\lambda}{\lambda + i - 1}G_0 + \frac{1}{\lambda + i - 1}\sum_{j<i}\delta_{\theta^j})$$

where $|H|$ is the total number of documents. By integrating over the unknown parameter $\theta$'s we can get

$$
\begin{aligned}
p(H) &= \int \left(\prod_{i=1}^{|H|}p(\mathbf{x}_i|\theta^i)\right) p(\theta^1, \theta^2, \ldots, \theta^{|H|})d\theta^1 d\theta^2 \ldots d\theta^{|H|} \\
&= \prod_{i=1}^{|H|}\left(\int p(\mathbf{x}_i|\theta^i)(\frac{\lambda}{\lambda + i - 1}G_0 + \frac{1}{\lambda + i - 1}\sum_{j<i}\delta_{\theta^j})d\theta^i\right) \qquad (3)
\end{aligned}
$$

Empirical Bayes method can be applied to equation (3) to estimate the model parameters by maximization[3]. In the following we discuss how to estimate parameters individually in detail.

### 3.1 Estimating $\pi_t$'s

Our hyperparameter $\pi$ vector contains $V$ number of parameters for the base distribution $G_0$, which can be treated as the expected distribution of $G$ – the prior of the cluster parameter $\theta$'s.

Although $\pi$ contains $V \approx 10^5$ number of actual parameters in our case, we can still use the empirical Bayes to do a reliable point estimation since the amount of data we have to represent general English is large (in our historical dataset there are around $10^6$ documents, around $1.8 \times 10^8$ English words in total) and highly informative about $\pi$. We use the smoothed estimation $\pi \propto (1 + n_1^{(H)}, 1 + n_2^{(H)}, \ldots, 1 + n_V^{(H)})$ where $n_t^{(H)} = \sum_{\mathbf{x} \in H} n_t^{(\mathbf{x})}$ is the total number of times that term $t$ happened in the collection $H$, and $\sum_{t=1}^{V} \pi_t$ should be normalized to 1. Furthermore, the pseudo-count one is added to alleviate the out-of-vocabulary problem.

### 3.2 Estimating $\gamma$

Though $\gamma$ is just a scalar parameter, it has the effect to control the uncertainty of the prior knowledge about how clusters are related to the general English model with the parameter $\pi$. We can see that $\gamma$ controls how far each new cluster can deviate from the general English model [4]. It can be estimated as follows:

$$\hat{\gamma} = \arg\max_{\gamma} \prod_{k=1}^{K} p(H_k|\gamma) = \arg\max_{\gamma} \prod_{k=1}^{K} \int p(H_k|\theta^{(k)})p(\theta^{(k)}|\gamma)d\theta^{(k)} \qquad (4)$$

$\hat{\gamma}$ can be numerically computed by solving the following equation:

$$K\Psi(\gamma) - K\sum_{v=1}^{V} \Psi(\gamma\pi_v)\pi_v + \sum_{k=1}^{K}\sum_{v=1}^{V} \Psi(\gamma\pi_v + n_v^{(H_k)})\pi_v - \sum_{k=1}^{K} \Psi(\gamma + \sum_{v=1}^{V} n_v^{(H_k)}) = 0$$

where the digamma function $\Psi(x)$ is defined as $\Psi(x) \equiv \frac{d}{dx}\ln\Gamma(x)$.

Alternatively we can choose $\gamma$ by evaluating over the historical dataset. This is applicable (though computationally expensive) since it is only a scalar parameter and we can pre-compute its possible range based on equation (4).

### 3.3 Estimating $\lambda$

The precision parameter $\lambda$ of the DP is also very important for the model, which controls how far the random distribution $G$ can deviate from the baseline model $G_0$. In our case, it is also the prior belief about how quickly new clusters will be generated in the sequence. Similarly we can use equation (3) to estimate $\lambda$, since items related to $\lambda$ can be factored out as $\prod_{i=1}^{|H|} \frac{\lambda^{y_i}}{\lambda+i-1}$. Suppose we have a labeled subset $H^L = \{(\mathbf{x}_1, y_1), (\mathbf{x}_2, y_2), \ldots, (\mathbf{x}_M, y_M)\}$ of training data, where $y_i$ is 1 if $\mathbf{x}_i$ is a novel document or 0 otherwise. Here we describe two possible choices:

1. The simplest way is to assume that $\lambda$ is a fixed constant during the process, and it can be computed as $\hat{\lambda} = \arg\max_{\lambda} \prod_{i \in H^L} \frac{\lambda^{y_t}}{\lambda+i-1}$, here $H^L$ denotes the subset of indices of labeled documents in the whole sequence.

2. The assumption that $\lambda$ is fixed maybe restrictive in reality, especially considering the fact that it reflects the generation rate of new clusters. More generally, we

can assume that $\lambda$ is some function of variable $i$. In particular, we assume $\lambda = a/i + b + ci$ where $a$, $b$ and $c$ are non-negative numbers. This formulation is a generalization of the above case, where the $i^{-1}$ term allows a much faster decrease at the beginning, and $c$ is the asymptotic rate of events happening as $i \rightarrow \infty$. Again the parameters $a$, $b$ and $c$ are estimated by MLE over the training dataset: $\hat{a}, \hat{b}, \hat{c} = \arg\max_{a,b,c>0} \prod_{i \in H^L} \frac{(a/i+b+ci)^{y_i}}{a/i+b+ci+i}$.

# 4 Experiments

We apply the above online clustering model to the novelty detection task in Topic Detection and Tracking (TDT). TDT has been a research community since its 1997 pilot study, which is a research initiative that aims at techniques to automatically process news documents in terms of events. There are several tasks defined in TDT, and among them Novelty Detection (a.k.a. First Story Detection or New Event Detection) has been regarded as the hardest task in this area [2]. The objective of the novelty detection task is to detect the earliest report for each event as soon as that report arrives in the temporal sequence of news stories.

## 4.1 Dataset

We use the TDT2 corpus as our historical dataset for estimating parameters, and use the TDT3 corpus to evaluate our model [5]. Notice that we have a subset of documents in the historical dataset (TDT2) for which events labels are given. The TDT2 corpus used for novelty detection task consists of 62,962 documents, among them 8,401 documents are labeled in 96 clusters. Stopwords are removed and words are stemmed, and after that there are on average 180 words per document. The total number of features (unique words) is around 100,000.

## 4.2 Evaluation Measure

In our experiments we use the standard TDT evaluation measure [1] to evaluate our results. The performance is characterized in terms of the probability of two types of errors: **miss** and **false alarm** ($P_{Miss}$ and $P_{FA}$). These two error probabilities are then combined into a single detection cost, $C_{det}$, by assigning costs to **miss** and **false alarm** errors:

$$C_{det} = C_{Miss} \cdot P_{Miss} \cdot P_{target} + C_{FA} \cdot P_{FA} \cdot P_{non-target}$$

where

1. $C_{Miss}$ and $C_{FA}$ are the costs of a miss and a false alarm, respectively,
2. $P_{Miss}$ and $P_{FA}$ are the conditional probabilities of a miss and a false alarm, respectively, and
3. $P_{target}$ and $P_{non-target}$ is the priori target probabilities ($P_{target} = 1 - P_{non-target}$).

It is the following normalized cost that is actually used in evaluating various TDT systems:

$$(C_{det})_{norm} = \frac{C_{det}}{\min(C_{Miss} \cdot P_{target}, C_{FA} \cdot P_{non-target})}$$

where the denominator is the minimum of two trivial systems. Besides, two types of evaluations are used in TDT, namely macro-averaged (topic-weighted) and micro-averaged

(story-weighted) evaluations. In macro-averaged evaluation, the cost is computed for every event, and then the average is taken. In micro-averaged evaluation the cost is averaged over all documents' decisions generated by the system, thus large event will have bigger impact on the overall performance. Note that macro-averaged evaluation is used as the primary evaluation measure in TDT. In addition to the binary decision "novel" or "non-novel", each system is required to generated a confidence score for each test document. The higher the score is, the more likely the document is novel. Here we mainly use the minimum cost to evaluate systems by varying the threshold, which is independent of the threshold setting.

## 4.3   Methods

One simple but effective method is the "GAC-INCR" clustering method [9] with cosine similarity metric and TFIDF term weighting, which has remained to be the top performing system in TDT 2002 & 2003 official evaluations. For this method the novelty confidence score we used is one minus the similarity score between the current cluster $\mathbf{x}_i$ and its nearest neighbor cluster: $s(\mathbf{x}_i) = 1.0 - \max_{j<i} sim(\mathbf{c}_i, \mathbf{c}_j)$, where $\mathbf{c}_i$ and $\mathbf{c}_j$ are the clusters that $\mathbf{x}_i$ and $\mathbf{x}_j$ are assigned to, respectively, and the similarity is taken to be the cosine similarity between two cluster vectors, where the ltc TFIDF term weighting scheme is used to scale each dimension of the vector. Our second method is to train a logistic regression model which combines multiple features generated by the GAC-INCR method. Those features not only include the similarity score used by the first method, but also include the size of its nearest cluster, the time difference between the current cluster and the nearest cluster, etc. We call this method "Logistic Regression", where we use the posterior probability $p(novelty|\mathbf{x}_i)$ as the confidence score. Finally, for our online clustering algorithm we choose the quantity $s(\mathbf{x}_i) = \log p(\mathbf{C}_0|\mathbf{x}_i)$ as the output confidence score.

## 4.4   Experimental Results

Our results for three methods are listed in Table 1, where both macro-averaged and micro-averaged minimum normalized costs are reported [6]. The GAC-INCR method performs very well, so does the logistic regression method. For our DP results, we observed that using the optimized $\hat{\gamma}$ will get results (not listed in the table) that are around 10% worse than using the $\gamma$ obtained through validation, which might be due to the flatness of the optimal function value as well as the sample bias of the clusters in the historical dataset[7]. Another observation is that the probabilistic decision does not actually improve the hard decision performance, especially for the $\lambda_{var}$ option. Generally speaking, our DP methods are comparable to the other two methods, especially in terms of topic-weighted measure.

Table 1: Results for novelty detection on TDT3 corpus

| Method | Topic-weighted Cost | Story-weighted Cost |
|---|---|---|
|  | COST (Miss, FA) | COST (Miss, FA) |
| GAC-INCR | 0.6945 (0.5614, 0.0272) | 0.7090 (0.5614, 0.0301) |
| Logistic Regression | 0.7027 (0.5732, 0.0264) | **0.6911** (0.5732, 0.0241) |
| DP with $\lambda_{fix}$, HD | 0.7054 (0.4737, 0.0473) | 0.7744 (0.5965, 0.0363) |
| DP with $\lambda_{var}$, HD | **0.6901** (0.5789, 0.0227) | 0.7541 (0.5789, 0.0358) |
| DP with $\lambda_{fix}$, PD | 0.7054 (0.4737, 0.0473) | 0.7744 (0.5965, 0.0363) |
| DP with $\lambda_{var}$, PD | 0.9025 (0.8772, 0.0052) | 0.9034 (0.8772, 0.0053) |

## 5 Related Work

Zaragoza et al. [11] applied a Bayesian Dirichlet-multinomial model to the ad hoc information retrieval task and showed that it is comparable to other smoothed language models. Blei et al. [3] used Chinese Restaurant Processes to model topic hierachies for a collection of documents. West et al. [8] discussed the sampling techniques for base distribution parameters in the Dirichlet process mixture model.

## 6 Conclusions and Future Work

In this paper we used a hierarchical probabilistic model for online document clustering. We modeled the generation of new clusters with a Dirichlet process mixture model, where the base distribution can be treated as the prior of general English model and the precision parameter is closely related to the generation rate of new clusters. Model parameters are estimated with empirical Bayes and validation over the historical dataset. Our model is evaluated on the TDT novelty detection task, and results show that our method is promising.

In future work we would like to investigate other ways of estimating parameters and use sampling methods to revisit previous cluster assignments. We would also like to apply our model to the retrospective detection task in TDT where systems do not need to make decisions online. Though its simplicity, the unigram multinomial model has its well-known limitation, which is the naive assumption about word independence. We also plan to explore richer but still tractable language models in this framework. Meanwhile, we would like to combine this model with the topic-conditioned framework [10] as well as incorporate hierarchical mixture model so that novelty detection will be conditioned on some topic, which will be modeled by either supervised or semi-supervised learning techniques.

## Footnotes

[1]For $\theta$ we use $\theta_v$ to denote the $v^{th}$ element in the vector, $\theta^i$ to denote the parameter vector that generates the $i^{th}$ document, and $\theta^{(j)}$ to denote the parameter vector for the $j^{th}$ cluster.

[2] Although documents are grouped into clusters in the historical dataset, we cannot make directly use of those labels due to the fact that clusters in the test dataset are different from those in the historical dataset.

[3] Since only a subset of documents are labeled in the historical dataset $H$, the maximization is only taken over the union of the labeled clusters.

[4]The mean and variance of a Dirichlet distribution $(\theta_1, \theta_2, \ldots, \theta_V) \sim \mathbf{Dir}(\gamma\pi_1, \gamma\pi_2, \ldots, \gamma\pi_V)$ are: $\mathbf{E}[\theta_v] = \pi_v$ and $\mathbf{Var}[\theta_v] = \frac{\pi_v(1-\pi_v)}{(\gamma+1)}$.

[5]Strictly speaking we only used the subsets of TDT2 and TDT3 that is designated for the novelty detection task.

[6]In TDT official evaluation there is also the DET curve, which is similar in spirit to the ROC curve that can reflects how the performance changes as the threshold varies. We will report those results in a longer version of this paper.

[7]It is known that the cluster labeling process of LDC is biased toward topics that will be covered in multiple languages instead of one single language.

## References

[1] The 2002 topic detection & tracking task definition and evaluation plan. *http://www.nist.gov/speech/tests/tdt/tdt2002/evalplan.htm*, 2002.

[2] Allan, J., Lavrenko, V. & Jin, H. First story detection in tdt is hard. In *Proc. of CIKM 2000.*

[3] Blei, D., Griffiths, T., Jordan, M. & Tenenbaum, J. Hierarchical topic models and the nested chinese restaurant process. *Advances in Neural Information Processing Systems*, 15, 2003.

[4] Ferguson, T. A Bayesian analysis of some nonparametric problems. *Annals of Statistics*, 1:209–230, 1973.

[5] Gelman A., Carlin, J., Stern, H. & Rubin, D. *Bayesian Data Analysis (2nd ed.).* CHAPMAN & HALL/CRC, 2003.

[6] Miller, D., Leek, T. & Schwartz, R. Bbn at trec 7: Using hidden markov models for information retrieval. In *TREC-7*, 1999.

[7] Minka, T. A family of algorithms for approximate Bayesian inference. *Ph.D. thesis, MIT, 2001.*

[8] West, M., Mueller, P. & Escobar, M.D. Hierarchical priors and mixture models, with application in regression and density estimation. In *Aspects of Uncertainty: A tribute to D. V. Lindley, A.F.M. Smith and P. Freeman, (eds.), Wiley, New York.*

[9] Yang, Y., Pierce, T. & Carbonell, J. A Study on Retrospective and On-line Event Detection. In *Proc. of SIGIR 1998.*

[10] Yang, Y., Zhang, J., Carnobell, J. & Jin, C. Topic-conditioned novelty detection. In *Proc. of 8th ACM SIGKDD International Conference on Knowledge Discovery and Data Mining*, 2002.

[11] Zaragoza, H., Hiemstra, D., Tipping, D. & Robertson, S. Bayesian extension to the language model for ad hoc information retrieval. In *Proc. SIGIR 2003.*
